# Extracting Rules from Artificial Neural Networks with Distributed Representations

**Sebastian Thrun**
University of Bonn
Department of Computer Science III
Römerstr. 164, D-53117 Bonn, Germany
E-mail: thrun@carbon.informatik.uni-bonn.de

## Abstract

Although artificial neural networks have been applied in a variety of real-world scenarios with remarkable success, they have often been criticized for exhibiting a low degree of human comprehensibility. Techniques that compile compact sets of symbolic rules out of artificial neural networks offer a promising perspective to overcome this obvious deficiency of neural network representations.

This paper presents an approach to the extraction of *if-then* rules from artificial neural networks. Its key mechanism is *validity interval analysis*, which is a generic tool for extracting symbolic knowledge by propagating rule-like knowledge through Backpropagation-style neural networks. Empirical studies in a robot arm domain illustrate the appropriateness of the proposed method for extracting rules from networks with real-valued and distributed representations.

## 1 Introduction

In the last few years artificial neural networks have been applied successfully to a variety of real-world problems. For example, neural networks have been successfully applied in the area of speech generation [12] and recognition [18], vision and robotics [8], handwritten character recognition [5], medical diagnostics [11], and game playing [13]. While in these and other approaches neural networks have frequently found to outperform more traditional approaches, one of their major shortcomings is their low degree of human comprehensibility.

In recent years, a variety of approaches for compiling rules out of networks have been proposed. Most approaches [1, 3, 4, 6, 7, 16, 17] compile networks into sets of rules with equivalent structure: Each processing unit is mapped into a separate rule–or a small set of rules–, and the ingoing weights are interpreted as preconditions to this rule. Sparse connectivity facilitates this type rule extraction, and so do binary activation values. In order to enforce such properties, which is a necessary prerequisite for these techniques to work effectively, some approaches rely on specialized training procedures, network initializations

and/or architectures.

While such a methodology is intriguing, as it draws a clear one-to-one correspondence between neural inference and rule-based inference, it is not universally applicable to arbitrary Backpropagation-style neural networks. This is because artificial neural networks might not meet the strong representational and structural requirements necessary for these techniques to work successfully. When the internal representation of the network is distributed in nature, individual hidden units typically do not represent clear, logical entities. One might argue that networks, if one is interested in extracting rules, should be constructed appropriately. But this would outrule most existing network implementations, as such considerations have barely played a role. In addition, such an argument would suppress the development of distributed, non-discrete internal representations, which have often be attributed for the generalization properties of neural networks. It is this more general class of networks that is at stake in this paper.

This paper presents a rule extraction method which finds rules by analyzing networks as a whole. The rules are of the type *"if x then y,"* where both $x$ and $y$ are described by a linear set of constraints. The engine for proving the correspondence of rule and network classification is VI-Analysis. Rules extracted by VI-Analysis can be proven to exactly describe the network.

## 2   Validity-Interval Analysis

Validity Interval Analysis (in short: VI-Analysis) is a generic tool for analyzing the input-output behavior of Backpropagation-style neural networks. In short, they key idea of VI-Analysis is to attach *intervals* to the activation range of each unit (or a subset of all units, like input and output units only), such that the network's activations *must* lie within these intervals. These intervals are called *validity intervals*. VI-Analysis checks whether such a set of intervals is *consistent*, *i.e.*, whether there exists a set of network activations inside the validity intervals. It does this by iteratively refining the validity intervals, excluding activations that are provably inconsistent with other intervals. In what follows we will present the general VI-Analysis algorithm, which can be found in more detail elsewhere [14].

Let $n$ denote the total number of units in the network, and let $x_i$ denote the *(output) activation* of unit $i$ ($i = 1, \ldots, n$). If unit $i$ is an input unit, its activation value will simply be the external input value. If not, *i.e.*, if $i$ refers to a hidden or an output unit, let $P(i)$ denote the set of units that are connected to unit $i$ through a link. The activation $x_i$ is computed in two steps:

$$x_i \;=\; \sigma_i(net_i) \qquad \text{with} \qquad net_i \;=\; \sum_{k \in P(i)} w_{ik} x_k + \theta_i$$

The auxiliary variable $net_i$ is the *net-input* of unit $i$, and $w_{ik}$ and $\theta_i$ are the *weights* and *biases*, respectively. $\sigma_i$ denotes the transfer function (squashing function), which usually is given by

$$\sigma_i(net_i) \;=\; \frac{1}{1 + e^{-net_i}} \quad \text{with} \quad \sigma_i^{-1}(x_i) \;=\; -\ln\left(\frac{1}{x_i} - 1\right)$$

Validity intervals for activation values $x_i$ are denoted by $[a_i, b_i]$. If necessary, validity intervals are projected into the net-input space of unit $i$, where they will be denoted by $[a_i', b_i']$. Let $\mathcal{I}$ be a set of validity intervals for (a subset of) all units. An activation vector $(x_1, \ldots, x_n)$ is said to be *admissible* with respect to $\mathcal{I}$, if all activations lie in $\mathcal{I}$. A set of intervals $\mathcal{I}$ is *consistent*, if there exists an admissible activation vector. Otherwise $\mathcal{I}$ is *inconsistent*.

Assume an initial set of intervals, denoted by $\mathcal{I}$, is given (in the next section we will present a procedure for generating initial intervals). VI-Analysis refines $\mathcal{I}$ iteratively using linear

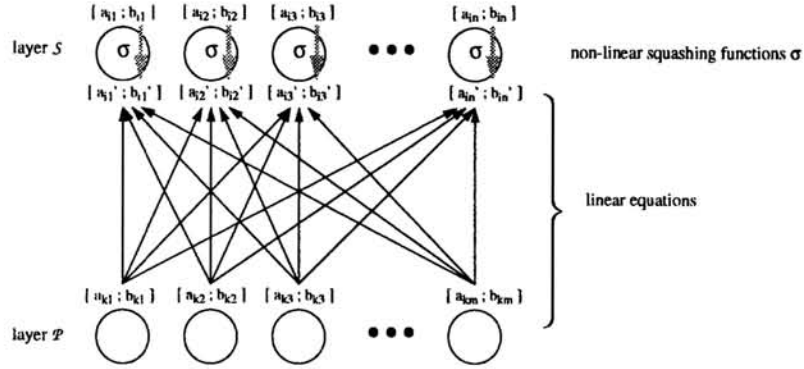

Figure 1: **VI-Analysis in a single weight layer.** Units in layer $\mathcal{P}$ are connected to the units in layer $\mathcal{S}$. A validity interval $[a_j, b_j]$ is assigned to each unit $j \in \mathcal{P} \cup \mathcal{S}$. By projecting the validity intervals for all $i \in \mathcal{S}$, intervals $[a'_i, b'_i]$ for the net-inputs $net_i$ are created. These, plus the validity intervals for all units $k \in \mathcal{P}$, form a set of linear constraints on the activations $x_k$ in layer $\mathcal{P}$. Linear programming is now employed to refine all interval bounds one-by-one.

---

programming [9], so that those activation values which are inconsistent with other intervals are excluded. In order to simplify the presentation, let us assume without loss of generality (a) that the network is layered and fully connected between two adjacent layers[1], and (b) that there is an interval $[a_i, b_i] \subseteq [0, 1]$ in $\mathcal{I}$ for every unit in $\mathcal{P}$ and $\mathcal{S}$.[2] Consider a single weight layer, connecting a layer of preceding units, denoted by $\mathcal{P}$, to a layer of succeeding units, denoted by $\mathcal{S}$ (*cf.* Fig. 1). In order to make linear programming techniques applicable, the non-linearity of the transfer function must be eliminated. This is achieved by projecting $[a_i, b_i]$ back to the corresponding net-input intervals[3] $[a'_i, b'_i] = \sigma^{-1}([a_i, b_i]) \in \mathfrak{R}^2$ for all $i \in \mathcal{S}$. The resulting validity intervals in $\mathcal{P}$ and $\mathcal{S}$ form the following set of linear constraints on the activation values in $\mathcal{P}$:

$$
\begin{aligned}
\forall k \in \mathcal{P}: &\quad x_k \geq a_k \quad \text{and} \quad x_k \leq b_k \\
\forall i \in \mathcal{S}: &\quad \sum_{k \in \mathcal{P}} w_{ik} x_k + \theta_i \geq a'_i \quad [\text{by substituting } net_i = \sum_{k \in \mathcal{P}} w_{ik} x_k + \theta_i] \\
&\quad \sum_{k \in \mathcal{P}} w_{ik} x_k + \theta_i \leq b'_i \quad [\text{by substituting } net_i = \sum_{k \in \mathcal{P}} w_{ik} x_k + \theta_i]
\end{aligned}
\tag{1}
$$

Notice that all these constraints are linear in the activation values $x_k$ ($k \in \mathcal{P}$). Linear programming allows to maximize or minimize arbitrary linear combinations of the variables $x_j$ while not violating a set of linear constraints [9]. Hence, linear programming can be applied to refine lower and upper bounds for validity intervals one-by-one.

In VI-Analysis, constraints are propagated in two phases:

1. **Forward phase.** To refine the bounds $a_i$ and $b_i$ for units $i \in \mathcal{S}$, new bounds $\hat{a}_i$ and $\hat{b}_i$ are

derived:
$$\hat{a}_i = \sigma(\hat{a}'_i) \quad \text{with} \quad \hat{a}'_i = \min net_i = \min \sum_{k \in \mathcal{P}} w_{ik} x_k + \theta_i$$

$$\hat{b}_i = \sigma(\hat{b}'_i) \quad \text{with} \quad \hat{b}'_i = \max net_i = \max \sum_{k \in \mathcal{P}} w_{ik} x_k + \theta_i$$

If $\hat{a}_i > a_i$, a tighter lower bound is found and $a_i$ is updated by $\hat{a}_i$. Likewise, $b_i$ is set to $\hat{b}_i$ if $\hat{b}_i < b_i$. Notice that the min/max operator is computed within the bounds imposed by Eq. 1, using the Simplex algorithm (linear programming) [9].

2. **Backward phase.** In the backward phase the bounds $a_k$ and $b_k$ of all units $k \in \mathcal{P}$ are refined.
$$\hat{a}_k = \min x_k \quad \text{and} \quad \hat{b}_k = \max x_k$$
As in the forward phase, $a_k$ is updated by $\hat{a}_k$ if $\hat{a}_k > a_k$, and $b_k$ is updated by $\hat{b}_k$ if $\hat{b}_k < b_k$.

If the network has multiple weight layers, this process is applied to all weight layers one-by-one. Repetitive refinement results in the propagation of interval constraints through multiple layers in both directions. The convergence of VI-Analysis follows from the fact that the update rule that intervals are changed monotonically, since they can only shrink or stay the same.

Recall that the "input" of VI-Analysis is a set of intervals $\mathcal{I} \subseteq [0, 1]^n$ that constrain the activations of the network. VI-Analysis generates a refined set of intervals, $\mathcal{I}' \subseteq \mathcal{I}$, so that all admissible activation values in the original intervals $\mathcal{I}$ are also in the refined intervals $\mathcal{I}'$. In other words, the difference between the original set of intervals and the refined set of intervals $\mathcal{I} - \mathcal{I}'$ is inconsistent.

In summary, VI-Analysis analyzes intervals $\mathcal{I}$ in order to detect inconsistencies. If $\mathcal{I}$ is found to be inconsistent, there is *provably* no admissible activation vector in $\mathcal{I}$. Detecting inconsistencies is the driving mechanism for the verification and extraction of rules presented in turn.

## 3   Rule Extraction

The rules considered in this paper are propositional *if-then rules*. Although VI-Analysis is able to prove rules expressed by arbitrary linear constraints [14], for the sake of simplicity we will consider only rules where the precondition is given by a set of intervals for the individual input values, and the output is a single target category. Rules of this type can be written as:

*If input* $\in$ *some hypercube* $\mathcal{I}$ *then class is* $C$    (or short: $\mathcal{I} \longrightarrow C$)

for some target class $C$.

The compliance of a rule with the network can be verified through VI-Analysis. Assume, without loss of generality, the network has a single output unit, and input patterns are classified as members of class $C$ if and only if the output activation, $x_{\text{out}}$, is larger than a threshold $c$ (see [14] for networks with multiple output units). A rule conjecture $\mathcal{I} \longrightarrow C$ is then verified by showing that there is no input vector $\vec{x} \in \mathcal{I}$ that falls into the opposite class, $\neg C$. This is done by including the (negated) condition $x_{\text{out}} \in [0, c]$ into the set of intervals: $\mathcal{I}_{\text{neg}} = \mathcal{I} + \{x_{\text{out}} \in [0, c]\}$. If the rule is correct, $x_{\text{out}}$ will never be in $[0, c]$. Hence, if VI-Analysis finds an inconsistency in $\mathcal{I}_{\text{neg}}$, the rule $\mathcal{I} \longrightarrow \neg C$ is proven to be incorrect, and thus the original rule $\mathcal{I} \longrightarrow C$ holds true for the network at hand. This illustrates how rules are *verified* using VI-Analysis.

It remains to be shown how such conjectures can be generated in a systematic way. Two major classes of approaches can be distinguished, *specific-to-general* and *general-to-specific*.

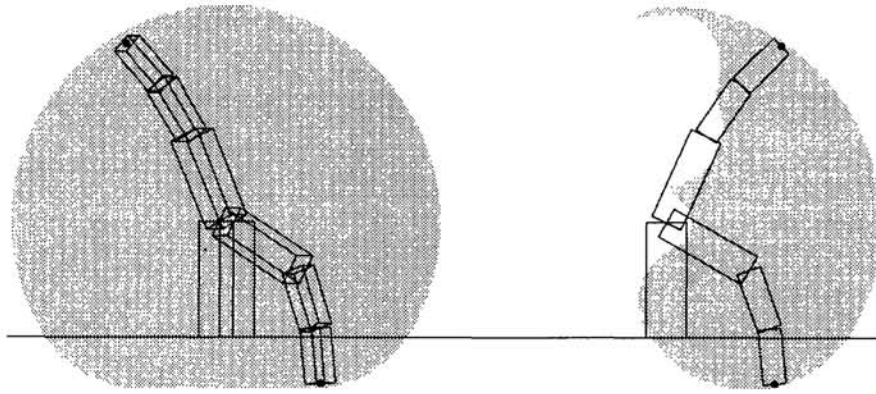

Figure 2: **Robot Arm.** (a) Front view of two arm configurations. (b) Two-dimensional side view. The grey area indicates the workspace, which partially intersects with the table.

1. **Specific-to-general.** A generic way to generate rules, which forms the basis for the experimental results reported in the next section, is to start with rather specific rules which are easy to verify, and gradually generalize those rules by enlarging the corresponding validity intervals. Imagine one has a training instance that, without loss of generality, falls into a class $C$. The input vector of the training instance already forms a (degenerate) set of validity intervals $\mathcal{I}$. VI-Analysis will, applied to $\mathcal{I}$, trivially confirm the membership in $C$, and hence the single-point rule $\mathcal{I} \longrightarrow C$. Starting with $\mathcal{I}$, a sequence of more general rule preconditions $\mathcal{I} \subset \mathcal{I}_1 \subset \mathcal{I}_2 \subset \ldots$ can be obtained by enlarging the precondition of the rule (*i.e.*, the input intervals $\mathcal{I}$) by small amounts, and using VI-Analysis to verify if the new rule is still a member of its class. In this way randomly generated instances can be used as "seeds" for rules, which are then generalized via VI-Analysis.

2. **General-to-specific.** An alternative way to extract rules, which has been studied in more detail elsewhere [14], works from general to specific. General-to-specific rule search maintains a list of non-proven conjectures, $R$. $R$ is initialized with the most general rules (like *"everything is in $C$"* and *"nothing is in $C$"*). VI-Analysis is then applied to prove rules in $R$. If it successfully confirms a rule, the rule and its complement is removed from $R$. If not, the rule is removed, too, but instead new rules are added to $R$. These new rules form a specialized version of the old rule, so that their disjunct is exactly the old rule. For example, new rules can be generated by splitting the hypercube spanned by the old rule into disjoint regions, one for each new rule. Then, the new set $R$ is checked with VI-Analysis. The whole procedure continues till $R$ is empty and the whole input domain is described by rules. In discrete domains, such a strategy amounts to searching directed acyclic graphs in breadth-first manner.

Obviously, there is a variety of alternative techniques to generate meaningful rule hypotheses. For example, one might employ a symbolic learning technique such as decision tree learning [10] to the same training data that was used for training the network. The rules, which are a result of the symbolic approach, constitute hypotheses that can be checked using VI-Analysis.

## 4 Empirical Results

In this section we will be interested in extracting rules in a real-valued robot arm domain. We trained a neural network to model the forward kinematics function of a 5 degree-of-freedom robot arm. The arm, a Mitsubishi RV-M1, is depicted in Fig. 2. Its kinematic function determines the position of the tip of the manipulator in $(x, y, z)$ workspace coordinates and

| coverage | average (per rule) | cumulative |
|---|---|---|
| first 10 rules | 9.79% | 30.2% |
| first 100 rules | 2.59% | 47.8% |
| first 1 000 rules | 1.20% | 61.6% |
| first 10 000 rules | 0.335 % | 84.4% |

Table 1: Rule coverage in the robot arm domain. These numbers include rules for both concepts, *SAFE* and *UNSAFE*.

---

the angle of the manipulator $h$ to the table based on the angles of the five joints. As can be seen in Fig. 2, the workspace intersects with the table on which the arm is mounted. Hence, some configurations of the joints are safe, namely those for which $z \geq 0$, while others can physically not be reached without a collision that would damage the robot (unsafe). When operating the robot arm one has to be able to tell safe from unsafe. Henceforth, we are interested in a set of rules that describes the subspace of safe and unsafe joint configurations.

A total of 8 192 training examples was used for training the network (four input, five hidden and four output units), resulting in a considerably accurate model of the kinematics of the robot arm. Notice that the network operates in a continuous space. Obviously, compiling the network into logical rules node-by-node, as frequently done in other approaches to rule extraction, is difficult due to the real-valued and distributed nature of the internal representation. Instead, we applied VI-Analysis using a specific-to-general mechanism as described above. More specifically, we incrementally constructed a collection of rules that gradually covered the workspace of the robot arm. Rules were generated whenever a (random) joint configuration was not covered by a previously generated rule. Table 1 shows average results that characterize the extraction of rules. Initially, each rule covers a rather large fraction of the 5-dimensional joint configuration space. As few as 11 rules, on average, suffice to cover more than 50% (by volume) of the whole input space. However, these 50% are the easy half. As the domain gets increasingly covered by rules, gradually more specific rules are generated in regions closer to the class boundary. After extracting 10,000 rules, only 84.4% of the input space is covered. Since the decision boundary between the two classes is highly non-linear, finitely many rules will never cover the input space completely.

How general are the rules extracted by VI-Analysis? Generally speaking, for joint configurations close to the class boundary, *i.e.*, where the tip of the manipulator is close to the table, we observed that the extracted rules were rather specific. If instead the initial configuration was closer to the center of a class, VI-Analysis was observed to produce more general rules that had a larger coverage in the workspace. Here VI-Analysis managed to extract surprisingly general rules. For example, the configuration $\vec{\alpha} = (30°, 80°, 20°, 60°, -20°)$, which is depicted in Fig. 3, yields the rule

*if* $\alpha_2 \leq 90.5°$ *and* $\alpha_3 \leq 27.3°$ *then*   *SAFE.*

Notice that out of 10 initial constraints, 8 were successfully removed by VI-Analysis. The rule lacks both bounds on $\alpha_1$, $\alpha_4$ and $\alpha_5$ and the lower bounds on $\alpha_2$ and $\alpha_3$. Fig. 3a shows the front view of the initial arm configuration and the generalized rule (grey area). Fig. 3b shows a side view of the arm, along with a slice of the rule (the base joint $\alpha_1$ is kept fixed). Notice that this very rule covers 17.1% of the configuration space (by volume). Such general rules were frequently found in the robot arm domain.

This concludes the brief description of the experimental results. Not mentioned here are results with different size networks, and results obtained for the MONK's benchmark problems. For example, in the MONK's problems [15], VI-Analysis successfully extracted compact target

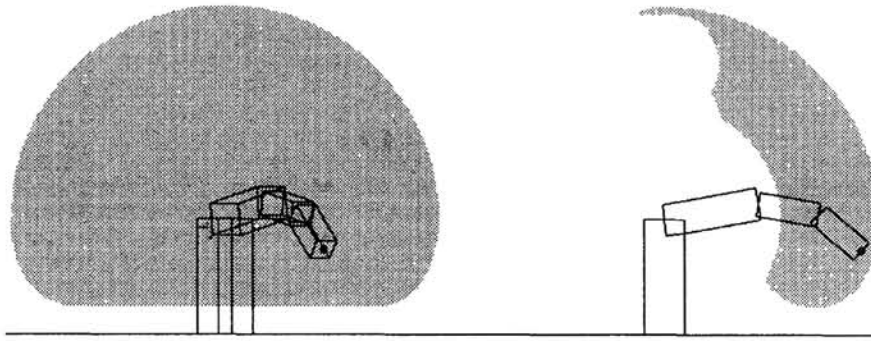

Figure 3: **A single rule, extracted from the network.** (a) Front view. (b) Two-dimensional side view. The grey area indicates safe positions for the tip of the manipulator.

concepts using the originally published weight sets. These results can be found in [14].

## 5 Discussion

In this paper we have presented a mechanism for the extraction of rules from Backpropagation-style neural networks. There are several limitations of the current approach that warrant future research. **(a) Speed.** While the one-to-one compilation of networks into rules is fast, rule extraction via VI-Analysis requires multiple runs of linear programming, each of which can be computationally expensive [9]. Searching the rule space without domain-specific search heuristics can thus be a most time-consuming undertaking. In all our experiments, however, we observed reasonably fast convergence of the VI-Algorithm, and we successfully managed to extract rules from larger networks in reasonable amounts of time. Recently, Craven and Shavlik proposed a more efficient search method which can be applied in conjunction with VI-Analysis [2]. **(b) Language.** Currently VI-Analysis is limited to the extraction of if-then rules with linear preconditions. While in [14] it has been shown how to generalize VI-Analysis to rules expressed by arbitrary linear constraints, a more powerful rule language is clearly desirable. **(c) Linear optimization.** Linear programming analyzes multiple weight layers independently, resulting in an overly careful refinement of intervals. This effect can prevent from detecting correct rules. If linear programming is replaced by a non-linear optimization method that considers multiple weight layers simultaneously, more powerful rules can be generated. On the other hand, efficient non-linear optimization techniques might find rules which do not describe the network accurately. Moreover, it is generally questionable whether there will ever exist techniques for mapping arbitrary networks accurately into compact rule sets. Neural networks are their own best description, and symbolic rules might not be appropriate for describing the input-output behavior of a complex neural network.

A key feature of of the approach presented in this paper is the particular way rules are extracted. Unlike other approaches to the extraction of rules, this mechanism does not compile networks into structurally equivalent set of rules. Instead it analyzes the input output relation of networks as a whole. As a consequence, rules can be extracted from unstructured networks with distributed and real-valued internal representations. In addition, the extracted rules describe the neural network accurately, regardless of the size of the network. This makes VI-Analysis a promising candidate for scaling rule extraction techniques to deep networks, in which approximate rule extraction methods can suffer from cumulative errors. We conjecture that such properties are important if meaningful rules are to be extracted in today's and tomorrow's successful Backpropagation applications.

## Acknowledgment
The author wishes to express his gratitude to Marc Craven, Tom Dietterich, Clayton McMillan, Tom Mitchell and Jude Shavlik for their invaluable feedback that has influenced this research.

## Footnotes

[1]This assumption simplifies the description of VI-Analysis, although VI-Analysis can also be applied to arbitrary non-layered, partially connected network architectures, as well as recurrent networks not examined here.

[2]The canonical interval $[0, 1]$ corresponds to the state of maximum ignorance about the activation of a unit, and hence is the default interval if no more specific interval is known.

[3]Here $\mathfrak{R}$ denotes the set of real numbers extended by $\pm\infty$. Notice that this projection assumes that the transfer function is monotonic.

## References

[1] M. W. Craven and J. W. Shavlik. Learning symbolic rules using artificial neural networks. In Paul E. Utgoff, editor, *Proceedings of the Tenth International Conference on Machine Learning*, 1993. Morgan Kaufmann.

[2] M. W. Craven and J. W. Shavlik. Using sampling and queries to extract rules from trained neural networks. In *Proceedings of the Eleventh International Conference on Machine Learning*, 1994. Morgan Kaufmann.

[3] L.-M. Fu. Integration of neural heuristics into knowledge-based inference. *Connection Science*, 1(3):325–339, 1989.

[4] C. L. Giles and C. W. Omlin. Rule refinement with recurrent neural networks. In *Proceedings of the IEEE International Conference on Neural Network*, 1993. IEEE Neural Network Council.

[5] Y. LeCun, B. Boser, J. S. Denker, D. Henderson, R. E. Howard, W. Hubbard, and L. D. Jackel. Backpropagation applied to handwritten zip code recognition. *Neural Computation*, 1:541–551, 1990.

[6] J. J. Mahoney and R. J. Mooney. Combining neural and symbolic learning to revise probabilistic rule bases. In J. E. Moody, S. J. Hanson, and R. P. Lippmann, editors, *Advances in Neural Information Processing Systems 5*, 1993. Morgan Kaufmann.

[7] C. McMillan, M. C. Mozer, and P. Smolensky. Rule induction through integrated symbolic and subsymbolic processing. In J. E. Moody, S. J. Hanson, and R. P. Lippmann, editors, *Advances in Neural Information Processing Systems 4*, 1992. Morgan Kaufmann.

[8] D. A. Pomerleau. ALVINN: an autonomous land vehicle in a neural network. Technical Report CMU-CS-89-107, Computer Science Dept. Carnegie Mellon University, Pittsburgh PA, 1989.

[9] W. H. Press. *Numerical recipes in C : the art of scientific computing*. Cambridge University Press, Cambridge [Cambridgeshire], New York, 1988.

[10] J. R. Quinlan. Induction of decision trees. *Machine Learning*, 1:81–106, 1986.

[11] J. Rennie. Cancer catcher: Neural net catches errors that slip through pap tests. *Scientific American*, 262, May 1990.

[12] T. J. Sejnowski and C. R. Rosenberg. Nettalk: A parallel network that learns to read aloud. Technical Report JHU/EECS-86/01, Johns Hopkins University, 1986.

[13] G. J. Tesauro. Practical issues in temporal difference learning. *Machine Learning*, 8, 1992.

[14] S. Thrun. Extracting provably correct rules from artificial neural networks. Technical Report IAI-TR-93-5, University of Bonn, Institut für Informatik III, D-53117 Bonn, May 1993.

[15] S. Thrun, J. Bala, E. Bloedorn, I. Bratko, B. Cestnik, J. Cheng, K. De Jong, S. Džeroski, D. Fisher, S. E. Fahlman, R. Hamann, K. Kaufman, S. Keller, I. Kononenko, J. Kreuziger, R. S. Michalski, T.M. Mitchell, P. Pachowicz, Y. Reich, H. Vafaie, W. Van de Welde, W. Wenzel, J. Wnek, and J. Zhang. The MONK's problems - a performance comparison of different learning algorithms. Technical Report CMU-CS-91-197, Carnegie Mellon University, Pittsburgh, PA, December 1991.

[16] G. Towell and J. W. Shavlik. Interpretation of artificial neural networks: Mapping knowledge-based neural networks into rules. In J. E. Moody, S. J. Hanson, and R. P. Lippmann, editors, *Advances in Neural Information Processing Systems 4*, 1992. Morgan Kaufmann.

[17] V. Tresp and J. Hollatz. Network structuring and training using rule-based knowledge. In J. E. Moody, S. J. Hanson, and R. P. Lippmann, editors, *Advances in Neural Information Processing Systems 5*, 1993. Morgan Kaufmann.

[18] A. H. Waibel. Modular construction of time-delay neural networks for speech recognition. *Neural Computation*, 1:39–46, 1989.
